# Postal Address Block Location Using A Convolutional Locator Network

**Ralph Wolf and John C. Platt**
Synaptics, Inc.
2698 Orchard Parkway
San Jose, CA 95134

## Abstract

This paper describes the use of a convolutional neural network to perform address block location on machine-printed mail pieces. Locating the address block is a difficult object recognition problem because there is often a large amount of extraneous printing on a mail piece and because address blocks vary dramatically in size and shape.

We used a convolutional locator network with four outputs, each trained to find a different corner of the address block. A simple set of rules was used to generate ABL candidates from the network output. The system performs very well: when allowed five guesses, the network will tightly bound the address delivery information in 98.2% of the cases.

## 1 INTRODUCTION

The U.S. Postal Service delivers about 350 million mail pieces a day. On this scale, even highly sophisticated and custom-built sorting equipment quickly pays for itself. Ideally, such equipment would be able to perform optical character recognition (OCR) over an image of the entire mail piece. However, such large-scale OCR is impractical given that the sorting equipment must recognize addresses on 18 mail pieces a second. Also, the large amount of advertising and other irrelevant text that can be found on some mail pieces could easily confuse or overwhelm the address recognition system. For both of these reasons, character recognition must occur

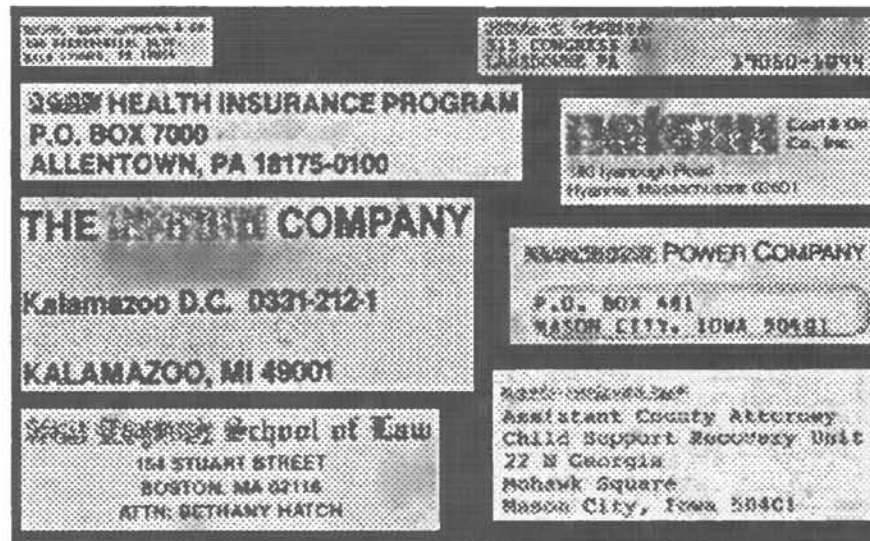

Figure 1: Typical address blocks from our data set. Notice the wide variety in the shape, size, justification and number of lines of text. Also notice the detached ZIP code in the upper right example. Note: The USPS requires us to preserve the confidentiality of the mail stream. Therefore, the name fields of all address block figures in this paper have been scrambled for publication. However, the network was trained and tested using unmodified images.

only on the relevant portion of the envelope: the destination address block. The system thus requires an address block location (ABL) module, which draws a tight bounding box around the destination address block.

The ABL problem is a challenging object recognition task because address blocks vary considerably in their size and shape (see figure 1). In addition, figures 2 and 3 show that there is often a great deal of advertising or other information on the mail piece which the network must learn to ignore.

Conventional systems perform ABL in two steps (Caviglione, 1990) (Palumbo, 1990). First, low-level features, such as blobs of ink, are extracted from the image. Then, address block candidates are generated using complex rules. Typically, there are hundreds of rules and tens of thousands of lines of code.

The architecture of our ABL system is very different from conventional systems. Instead of using low-level features, we train a neural network to find high-level abstract features of an address block. In particular, our neural network detects the corners of the bounding box of the address block. By finding abstract features instead of trying to detect the whole address block in one step, we build a large degree of scale and shape invariance into the system. By using a neural network, we do not need to develop explicit rules or models of address blocks, which yields a more accurate system.

Because the features are high-level, it becomes easy to combine these features into object hypotheses. We use simple address block statistics to convert the corner features into object hypotheses, using only 200 lines of code.

## 2 SYSTEM ARCHITECTURE

Our ABL system takes 300 dpi grey scale images as input and produces a list of the 5 most likely ABL candidates as output. The system consists of three parts: the preprocessor, a convolutional locator network, and a candidate generator.

### 2.1 PREPROCESSOR

The preprocessor serves two purposes. First, it substantially reduces the resolution of the input image, therefore decreasing the computational requirements of the neural network. Second, the preprocessor enhances spatial frequencies in the image which are associated with address text. The recipe used for the preprocessing is as follows:

```
1: Clip the top 20% of the image.
2: Spatially filter with a passband of 0.3 to 1.4mm.
3: Take the absolute value of each pixel.
4: Low-pass filter and subsample by a factor of 16 in X and Y.
5: Perform a linear contrast stretch, mapping the darkest
   pixel to 1.0 and the lightest pixel to 0.0.
```

The effect of this preprocessing can be seen in figures 2 and 3.

### 2.2 CONVOLUTIONAL LOCATOR NETWORK

We use a convolutional locator network (CLN) to find the corners of the bounding box. Each layer of a CLN convolves its weight pattern in two dimensions over the outputs of the previous layer (LeCun, 1989) (Fukushima, 1980). Unlike standard convolutional networks, the output of a CLN is a set of images, in which regions of activity correspond to recognition of a particular object. We train an output neuron of a CLN to be on when the receptive field of that neuron is over an object or feature, and off everywhere else.

CLNs have been previously used to assist in the segmentation step for optical character recognition, where a neuron is trained to turn on in the center of every character, regardless of the identity of the character (Martin, 1992) (Platt, 1992). The recognition of an address block is a significantly more difficult image segmentation problem because address blocks vary over a much wider range than printed characters (see figure 1).

The output of the CLN is a set of four feature maps, each corresponding to one corner of the address block. The intensity of a pixel in a given feature map represents the likelihood that the corresponding corner of the address block is located at that pixel.

Figure 4 shows the architecture of our convolutional locator network (CLN). It has three layers of trainable weights, with a total of 22,800 free parameters. The network was trained via weight-shared backpropagation. The network was trained for 23 epochs on 800 mail piece images. This required 125 hours of cpu-time on an i860 based computer. Cross validation and final testing was done with two additional

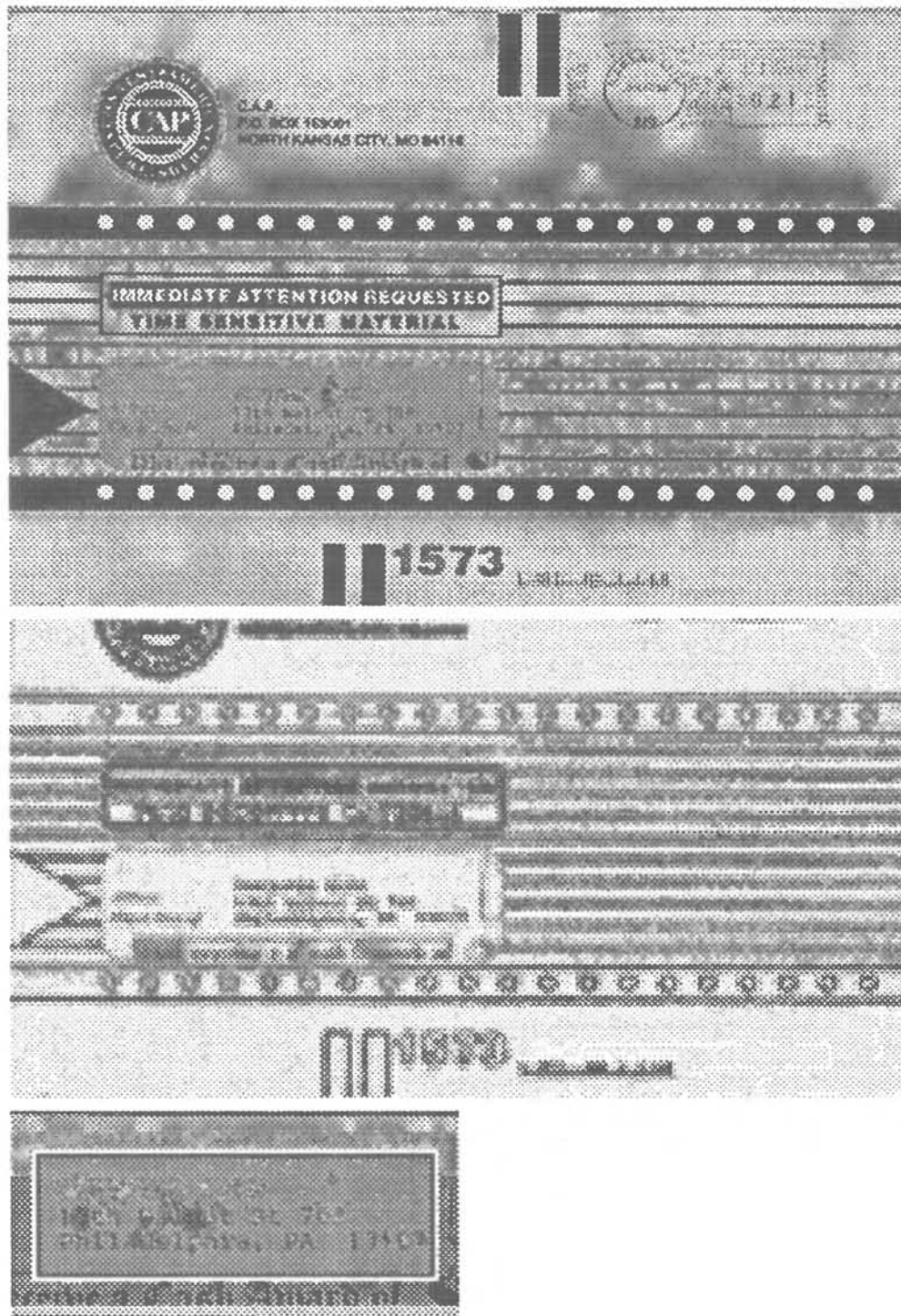

Figure 2: The network operating on an example from the test set. The top image is the original image. The middle image is the image that is fed to the CLN after preprocessing. The preprocessing enhances the text and suppresses the background color. The bottom image is the first candidate of the ABL system. The output of the system is shown with a white and black rectangle. In this case, the first candidate is correct. Notice that our ABL system does not get confused by the horizontal lines in the image, which would confound a line-finding-based ABL system.

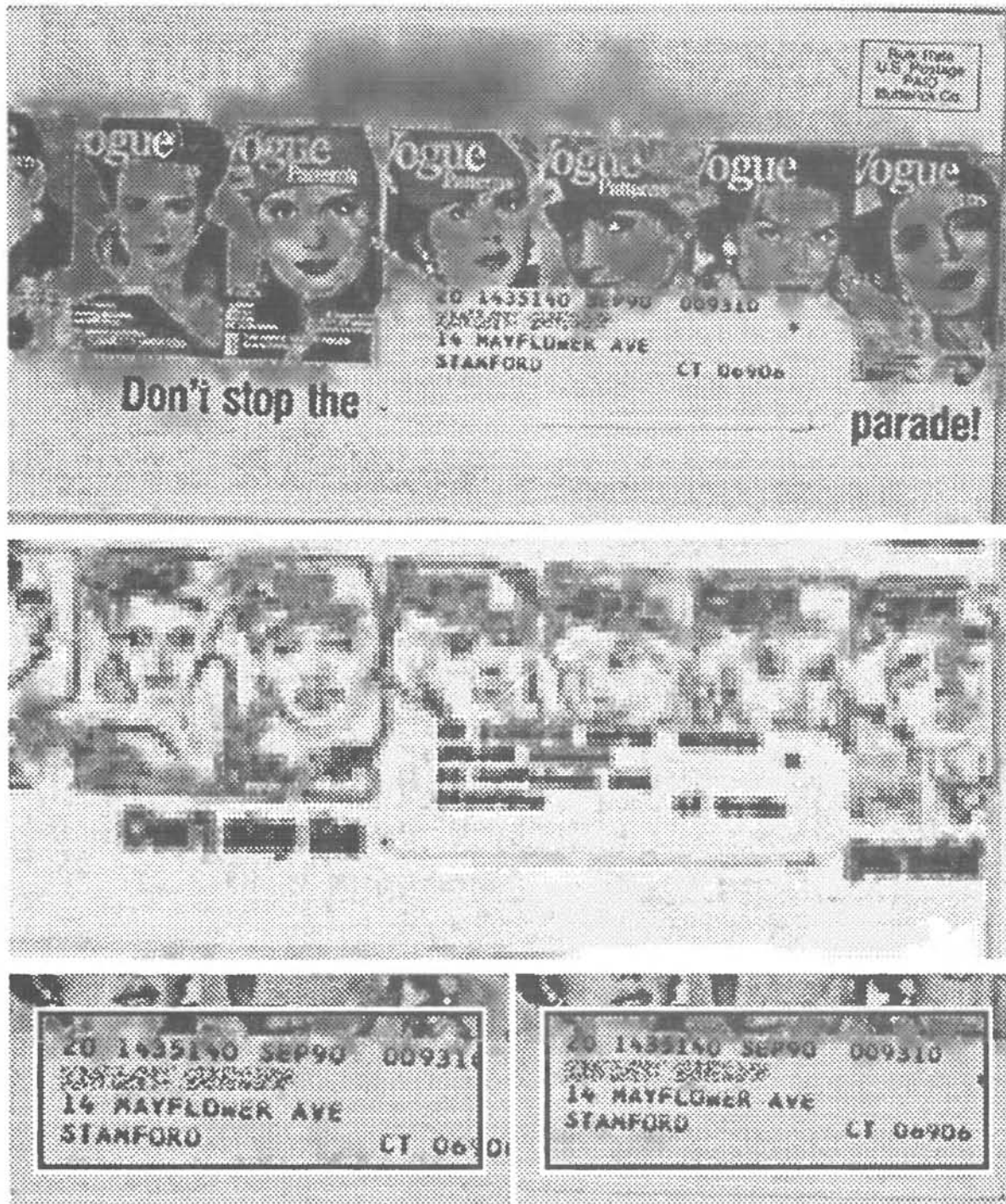

Figure 3: Another example from the test set. The preprocessed image still has a large amount of background noise. In this example, the first candidate of the ABL system (shown in the lower left) was almost correct, but the ZIP code got truncated. The second candidate of the system (shown in the lower right) gives the complete address.

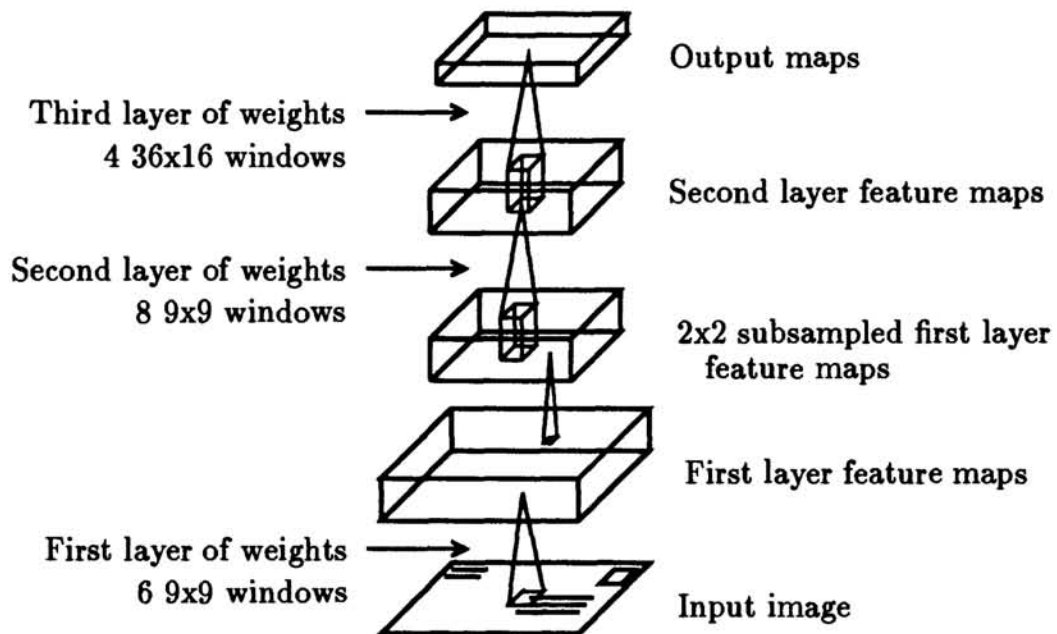

Third layer of weights
4 36x16 windows

Second layer of weights
8 9x9 windows

First layer of weights
6 9x9 windows

Output maps

Second layer feature maps

2x2 subsampled first layer
feature maps

First layer feature maps

Input image

Figure 4: The architecture of the convolutional locator network used in our ABL system.

data sets of 500 mail pieces each. All together, these 1800 images represent 6 Gbytes of raw data, or 25 Mbytes of preprocessed images.

## 2.3  CANDIDATE GENERATOR

The candidate generator uses the following recipe to convert the output maps of the CLN into a list of ABL candidates:

    1: Find the top 10 local maxima in each feature map.
    2: Construct all possible ABL candidates by combining pairs of
       local maxima from opposing corners.
    3: Discard candidates which have negative length or width.
    4: Compute confidence of each candidate.
    5: Sort the candidates according to confidence.
    6: Remove duplicate and near duplicate candidates.
    7: Pad the candidates by a fixed amount on all sides.

The confidence of an address block candidate is:

$$C_{\text{address block}} = P_{\text{size}} P_{\text{location}} \prod_{i=1}^{2} C_i$$

where $C_{\text{address block}}$ is the confidence of the address block candidate, $P_{\text{size}}$ is the prior probability of finding an address block of the hypothesized size, $P_{\text{location}}$ is the prior probability of finding an address block in the hypothesized location, and

$C_i$ are the value of each of the output maxima. The prior probabilities $P_{size}$ and $P_{location}$ were based on smoothed histograms generated from the training set and validation set truths.

Steps 6 and 7 each contain 4 tuning parameters which we optimized using the validation set and then froze before evaluating the final test set.

## 3   SYSTEM PERFORMANCE

Figures 2 and 3 show the performance of the system on two challenging mail pieces from the final test set. We examined and classified the response of the system to all 500 test images. When allowed to produce five candidates, the ABL system found 98.2% of the address blocks in the test images.

More specifically, 96% of the images have a compact bounding box for the complete address block. Another 2.2% have bounding boxes which contain all of the delivery information, but omit part of the name field. The remaining 1.8% fail, either because none of the candidates contain all the delivery information, or because they contain too much non-address information. The average number of candidates required to find a compact bounding box is only 1.4.

## 4   DISCUSSION

This paper demonstrates that using a CLN to find abstract features of an object, rather than locating the entire object, provides a reasonable amount of insensitivity to the shape and scale of the object. In particular, the completely identified address blocks in the final test set had aspect ratios which ranged from 1.3 to 6.1 and their absolute X and Y dimensions both varied over a 3:1 range. They contained anywhere from 2 to 6 lines of text.

In the past, rule-based systems for object recognition were designed from scratch and required a great deal of domain-specific knowledge. CLNs can be trained to recognize different classes of objects without a lot of domain-specific knowledge. Therefore, CLNs are a general purpose object segmentation and recognition architecture.

The basic computation of a CLN is a high-speed convolution, which can be cost-effectively implemented by using parallel hardware (Säckinger, 1992). Therefore, CLNs can be used to reduce the complexity and cost of hardware recognition systems.

## 5   CONCLUSIONS

In this paper, we have described a software implementation for an address block location system which uses a convolutional locator network to detect the corners of the destination address on machine printed mail pieces.

The success of this system suggests a general approach to object recognition tasks where the objects vary considerably in size and shape. We suggest the following

three-step approach: use a simple preprocessing algorithm to enhance stimuli which are correlated to the object, use a CLN to detect abstract features of the objects in the preprocessed image, and construct object hypotheses by a simple analysis of the network output. The use of CLNs to detect abstract features enables versatile object recognition architectures with a reasonable amount of scale and shape invariance.

## Acknowledgements

This work was funded by USPS Contract No. 104230-90-C-3441. The authors would like to thank Dr. Binh Phan of the USPS for his generous advice and encouragement. The images used in this work were provided by the USPS.

## References

Caviglione, M., Scaiola, (1990), "A Modular Real-time Vision System for Address Block Location," *Proc. 4th Advanced Technology Conference*, USPS, 42–56.

Fukushima, K., (1980), "Neocognitron: A Self-Organizing Neural Network Model for a Mechanism of Pattern Recognition Unaffected by Shift in Position." *Biological Cybernetics*, **36**, 193–202.

LeCun, Y., Boser, B., Denker, J.S., Henderson, D., Howard, R. E., Hubbard, W., Jackel, L. D., (1989), "Backpropagation Applied to Handwritten Zip Code Recognition" *Neural Computation*, **1**, 541–551.

Martin, G., Rashid, M., (1992), "Recognizing Overlapping Hand-Printed Characters by Centered-Object Integrated Segmentation and Recognition," *Advances in Neural Information Processing Systems*, **4**, 504–511.

Palumbo, P. W., Soh, J., Srihari, S. N., Demjanenjo, V., Sridhar, R., (1990), "Real-Time Address Block Location using Pipelining and Multiprocessing," *Proc. 4th Advanced Technology Conference*, USPS, 73–87.

Platt, J., Decker, J. E, LeMoncheck, J. E., (1992), "Convolutional Neural Networks for the Combined Segmentation and Recognition of Machine Printed Characters," *Proc. 5th Advanced Technology Conference*, USPS, 701–713.

Säckinger, E., Boser, B., Bromley, J., LeCun, Y., Jackel, L., (1992) "Application of the ANNA neural network chip to high-speed character recognition," *IEEE Trans. Neural Networks*, **3**, (3), 498–505.